# Active Preference Learning with Discrete Choice Data

**Eric Brochu, Nando de Freitas and Abhijeet Ghosh**
Department of Computer Science
University of British Columbia
Vancouver, BC, Canada
`{ebrochu, nando, ghosh}@cs.ubc.ca`

## Abstract

We propose an active learning algorithm that learns a continuous valuation model from discrete preferences. The algorithm automatically decides what items are best presented to an individual in order to find the item that they value highly in as few trials as possible, and exploits quirks of human psychology to minimize time and cognitive burden. To do this, our algorithm maximizes the expected improvement at each query without accurately modelling the entire valuation surface, which would be needlessly expensive. The problem is particularly difficult because the space of choices is infinite. We demonstrate the effectiveness of the new algorithm compared to related active learning methods. We also embed the algorithm within a decision making tool for assisting digital artists in rendering materials. The tool finds the best parameters while minimizing the number of queries.

## 1 Introduction

*A computer graphics artist sits down to use a simple renderer to find appropriate surfaces for a typical reflectance model. It has a series of parameters that must be set to control the simulation: "specularity", "Fresnel reflectance coefficient", and other, less-comprehensible ones. The parameters interact in ways difficult to discern. The artist knows in his mind's eye what he wants, but he's not a mathematician or a physicist — no course he took during his MFA covered Fresnel reflectance models. Even if it had, would it help? He moves the specularity slider and waits for the image to be generated. The surface is too shiny. He moves the slider back a bit and runs the simulation again. Better. The surface is now appropriately dull, but too dark. He moves a slider down. Now it's the right colour, but the specularity doesn't look quite right any more. He repeatedly bumps the specularity back up, rerunning the renderer at each attempt until it looks right. Good. Now, how to make it look metallic...?*

Problems in simulation, animation, rendering and other areas often take such a form, where the desired end result is identifiable by the user, but parameters must be tuned in a tedious trial-and-error process. This is particularly apparent in psychoperceptual models, where continual tuning is required to make something "look right". Using the animation of character walking motion as an example, for decades, animators and scientists have tried to develop objective functions based on kinematics, dynamics and motion capture data [Cooper *et al.*, 2007]. However, even when expensive mocap is available, we simply have to watch an animated film to be convinced of how far we still are from solving the gait animation problem. Unfortunately, it is not at all easy to find a mapping from parameterized animation to psychoperceptual plausibility. *The perceptual objective function is simply unknown.* Fortunately, however, it is fairly easy to judge the quality of a walk — in fact, it is trivial and almost instantaneous. The application of this principle to animation and other psychoperceptual tools is motivated by the observation that humans often seem to be forming a mental model of the objective function. This model enables them to *exploit* feasible regions of the parameter space where the valuation is predicted to be high and to *explore* regions of high uncertainty. It is our the-

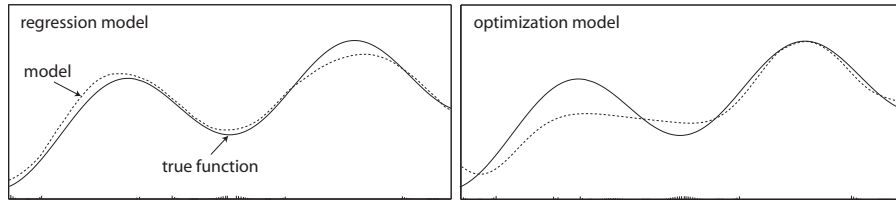

Figure 1: An illustrative example of the difference between models learned for regression vesus optimization. The regression model fits the true function better overall, but doesn't fit at the maximum better than anywhere else in the function. The optimization model is less accurate overall, but fits the area of the maximum very well. When resources are limited, such as an active learning environment, it is far more useful to fit the area of interest well, even at the cost of overall predictive performance. Getting a good fit for the maximum will require many more samples using conventional regression.

sis that the process of tweaking parameters to find a result that looks "right" is akin to sampling a perceptual objective function, and that twiddling the parameters to find the best result is, in essence, optimization. Our objective function is the psycho-perceptual process underlying judgement — how well a realization fits what the user has in mind. Following the econometrics terminology, we refer to the objective as the *valuation*. In the case of a human being rating the suitability of a simulation, however, it is not possible to evaluate this function over the entire domain. In fact, it is in general impossible to even sample the function directly and get a consistent response! While it would theoretically be possible to ask the user to rate realizations with some numerical scale, such methods often have problems with validity and reliability. Patterns of use and other factors can result in a *drift effect*, where the scale varies over time [Siegel and Castellan, 1988]. However, human beings *do* excel at comparing options and expressing a preference for one over others [Kingsley, 2006]. This insight allows us to approach the optimization function in another way. By presenting two or more realizations to a user and requiring only that they indicate preference, we can get far more robust results with much less cognitive burden on the user [Kendall, 1975]. While this means we can't get responses for a valuation function directly, we model the valuation as a latent function, inferred from the preferences, which permits an active learning approach [Cohn *et al.*, 1996; Tong and Koller, 2000].

This motivates our second major insight — *it is not necessary to accurately model the entire objective function*. The problem is actually one of optimization, not regression (Figure 1). We can't directly maximize the valuation function, so we propose to use an *expected improvement function* (EIF) [Jones *et al.*, 1998; Sasena, 2002]. The EIF produces an estimate of the utility of knowing the valuation at any point in the space. The result is a principled way of trading off exploration (showing the user examples unlike any they have seen) and exploitation (trying to show the user improvements on examples they have indicated preference for). Of course, regression-based learning can produce an accurate model of the entire valuation function, which would also allow us to find the best valuation. However, this comes at the cost of asking the user to compare many, many examples that have no practical relation what she is looking for, as we demonstrate experimentally in Sections 3 and 4. Our method tries instead to make the most efficient possible use of the user's time and cognitive effort.

Our goal is to exploit the strengths of human psychology and perception to develop a novel framework of valuation optimization that uses active preference learning to find the point in a parameter space that approximately maximizes valuation with the least effort to the human user. Our goal is to offload the cognitive burden of estimating and exploring different sets of parameters, though we can incorporate "slider twiddling" into the framework easily. In Section 4, we present a simple, but practical application of our model in a material design gallery that allows artists to find particular appearance rendering effects. Furthermore, the valuation function can be *any* psychoperceptual process that lends itself to sliders and preferences: the model can support an animator looking for a particular "cartoon physics" effect, an artist trying to capture a particular mood in the lighting of a scene, or an electronic musician looking for a specific sound or rhythm. Though we use animation and rendering as motivating domains, our work has a broad scope of application in music and other arts, as well as psychology, marketing and econometrics, and human-computer interfaces.

## 1.1 Previous Work

Probability models for learning from discrete choices have a long history in psychology and econometrics [Thurstone, 1927; Mosteller, 1951; Stern, 1990; McFadden, 2001]. They have been studied extensively for use in rating chess players, and the Elo system [Élő, 1978] was adopted by the World Chess Federation FIDE to model the probability of one player defeating another. Glickman and Jensen [2005] use Bayesian optimal design for adaptively finding pairs for tournaments. These methods all differ from our work in that they are intended to predict the probability of a preference outcome over a finite set of possible pairs, whereas we work with infinite sets and are only incidentally interested in modelling outcomes.

In Section 4, we introduce a novel "preference gallery" application for designing simulated materials in graphics and animation to demonstrate the practical utility of our model. In the computer graphics field, the *Design Gallery* [Marks *et al.*, 1997] for animation and the gallery navigation interface for Bidirectional Reflectance Distribution Functions (BRDFs) [Ngan *et al.*, 2006] are artist-assistance tools most like ours. They both uses non-adaptive heuristics to find the set of input parameters to be used in the generation of the display. We depart from this heuristic treatment and instead present a principled probabilistic decision making approach to model the design process.

Parts of our method are based on [Chu and Ghahramani, 2005b], which presents a preference learning method using probit models and Gaussian processes. They use a Thurstone-Mosteller model, but with an innovative nonparametric model of the valuation function. [Chu and Ghahramani, 2005a] adds active learning to the model, though the method presented there differs from ours in that realizations are selected from a finite pool to maximize informativeness. More importantly, though, this work, like much other work in the field [Seo *et al.*, 2000; Guestrin *et al.*, 2005], is concerned with learning the entire latent function. As our experiments show in Section 3, this is too expensive an approach for our setting, leading us to develop the new active learning criteria presented here.

## 2 Active Preference Learning

By querying the user with a paired comparison, one can estimate statistics of the valuation function at the query point, but only at considerable expense. Thus, we wish to make sure that the samples we do draw will generate the maximum possible improvement.

Our method for achieving this goal iterates the following steps:

1. **Present the user with a new pair and record the choice**: Augment the training set of paired choices with the new user data.

2. **Infer the valuation function**: Here we use a Thurstone-Mosteller model with Gaussian processes. See Sections 2.1 and 2.2 for details. *Note that we are not interested in predicting the value of the valuation function over the entire feasible domain, but rather in predicting it well near the optimum.*

3. **Formulate a statistical measure for exploration-exploitation**: We refer to this measure as the expected improvement function (EIF). Its maximum indicates where to sample next. EI is a function of the Gaussian process predictions over the feasible domain. See Section 2.3.

4. **Optimize the expected improvement function to obtain the next query point**: Finding the maximum of the EI corresponds to a constrained nonlinear programming problem. See Section 2.3.

### 2.1 Preference Learning Model

Assume we have shown the user $M$ pairs of items. In each case, the user has chosen which item she likes best. The dataset therefore consists of the ranked pairs $\mathcal{D} = \{\mathbf{r}_k \succ \mathbf{c}_k; \ k = 1, \ldots, M\}$, where the symbol $\succ$ indicates that the user prefers $\mathbf{r}$ to $\mathbf{c}$. We use $\mathbf{x}_{1:N} = \{\mathbf{x}_1, \mathbf{x}_2, \ldots, \mathbf{x}_N\}, \mathbf{x}_i \in \mathcal{X} \subseteq \mathbb{R}^d$, to denote the $N$ elements in the training data. That is, $\mathbf{r}_k$ and $\mathbf{c}_k$ correspond to two elements of $\mathbf{x}_{1:N}$.

Our goal is to compute the item $\mathbf{x}$ (not necessarily in the training data) with the highest user valuation in as few comparisons as possible. We model the valuation functions $u(\cdot)$ for $\mathbf{r}$ and $\mathbf{c}$ as follows:

$$\begin{aligned} u(\mathbf{r}_k) &= f(\mathbf{r}_k) + e_{rk} \\ u(\mathbf{c}_k) &= f(\mathbf{c}_k) + e_{ck}, \end{aligned} \tag{1}$$

where the noise terms are Gaussian: $e_{rk} \sim \mathcal{N}(0, \sigma^2)$ and $e_{ck} \sim \mathcal{N}(0, \sigma^2)$. Following [Chu and Ghahramani, 2005b], we assign a nonparametric Gaussian process prior to the unknown mean valuation: $f(\cdot) \sim GP(0, K(\cdot, \cdot))$. That is, at the $N$ training points. $p(\mathbf{f}) = |2\pi\mathbf{K}|^{-\frac{1}{2}} \exp\left(-\frac{1}{2}\mathbf{f}^T\mathbf{K}^{-1}\mathbf{f}\right)$, where $\mathbf{f} = \{f(\mathbf{x}_1), f(\mathbf{x}_2), \ldots, f(\mathbf{x}_N)\}$ and the symmetric positive definite covariance $\mathbf{K}$ has entries (kernels) $\mathbf{K}_{ij} = k(\mathbf{x}_i, \mathbf{x}_j)$. Initially we learned these parameters via maximum likelihood, but soon realized that this was unsound due to the scarcity of data. To remedy this, we elected to use subjective priors using simple heuristics, such as expected dataset spread. Although we use Gaussian processes as a principled method of modelling the valuation, other techniques, such as wavelets could also be adopted.

Random utility models such as (1) have a long and influential history in psychology and the study of individual choice behaviour in economic markets. Daniel McFadden's Nobel Prize speech [McFadden, 2001] provides a glimpse of this history. Many more comprehensive treatments appear in classical economics books on discrete choice theory.

Under our Gaussian utility models, the probability that item $\mathbf{r}$ is preferred to item $\mathbf{c}$ is given by:

$$P(\mathbf{r}_k \succ \mathbf{c}_k) = P(u(\mathbf{r}_k) > u(\mathbf{c}_k)) = P(e_{ck} - e_{rk} < f(\mathbf{r}_k) - f(\mathbf{c}_k)) = \Phi\left[\frac{f(\mathbf{r}_k) - f(\mathbf{c}_k)}{\sqrt{2}\sigma}\right],$$

where $\Phi(d_k) = \frac{1}{\sqrt{2\pi}}\int_{-\infty}^{d_k} \exp\left(-a^2/2\right) da$ is the cumulative function of the standard Normal distribution. This model, relating binary observations to a continuous latent function, is known as the Thurstone-Mosteller law of comparative judgement [Thurstone, 1927; Mosteller, 1951]. In statistics it goes by the name of binomial-probit regression. Note that one could also easily adopt a logistic (sigmoidal) link function $\varphi(d_k) = (1 + \exp(-d_k))^{-1}$. In fact, such choice is known as the Bradley-Terry model [Stern, 1990]. If the user had more than two choices one could adopting a multinomial-probit model. This multi-category extension would, for example, enable the user to state no preference for any of the two items being presented.

## 2.2 Inference

Our goal is to estimate the posterior distribution of the latent utility function given the discrete data. That is, we want to compute $p(\mathbf{f}|\mathcal{D}) \propto p(\mathbf{f})\prod_{k=1}^{M} p(d_k|\mathbf{f})$, where $d_k = \frac{f(\mathbf{r}_k) - f(\mathbf{c}_k)}{\sqrt{2}\sigma}$. Although there exist sophisticated variational and Monte Carlo methods for approximating this distribution, we favor a simple strategy: Laplace approximation. Our motivation for doing this is the simplicity and computational efficiency of this technique. Moreover, given the amount of uncertainty in user valuations, we believe the choice of approximating technique plays a small role and hence we expect the simple Laplace approximation to perform reasonably in comparison to other techniques. The application of the Laplace approximation is fairly straightforward, and we refer the reader to [Chu and Ghahramani, 2005b] for details.

Finally, given an arbitrary test pair, the predicted utility $f^\star$ and $\mathbf{f}$ are jointly Gaussian. Hence, one can obtain the conditional $p(f^\star|\mathbf{f})$ easily. Moreover, the predictive distribution $p(f^\star|\mathcal{D})$ follows by straightforward convolution of two Gaussians: $p(f^\star|\mathcal{D}) = \int p(f^\star|\mathbf{f})p(\mathbf{f}|\mathcal{D})d\mathbf{f}$. One of the criticisms of Gaussian processes, the fact that they are slow with large data sets, is not a problem for us, since active learning is designed explicitly to minimize the number of training data.

## 2.3 The Expected Improvement Function

Now that we are armed with an expression for the predictive distribution, we can use it to decide what the next query should be. In loose terms, the predictive distribution will enable us to balance the tradeoff of exploiting and exploring. When exploring, we should choose points where the predicted variance is large. When exploiting, we should choose points where the predicted mean is large (high valuation).

Let $\mathbf{x}^\star$ be an arbitrary new instance. Its predictive distribution $p(f^\star(\mathbf{x}^\star)|\mathcal{D})$ has sufficient statistics $\{\mu(\mathbf{x}^\star) = \mathbf{k}^{\star T}\mathbf{K}^{-1}\mathbf{f}^{MAP}, s^2(\mathbf{x}^\star) = \mathbf{k}^{\star\star} - \mathbf{k}^{\star T}(\mathbf{K} + \mathbf{C}_{MAP}^{-1})^{-1}\mathbf{k}^\star\}$, where, now, $\mathbf{k}^{\star T} = [k(\mathbf{x}^\star, \mathbf{x}_1)\cdots k(\mathbf{x}^\star, \mathbf{x}_N)]$ and $\mathbf{k}^{\star\star} = k(\mathbf{x}^\star, \mathbf{x}^\star)$. Also, let $\mu_{\max}$ denote the highest estimate of the predictive distribution thus far. That is, $\mu_{\max}$ is the highest valuation for the data provided by the individual.

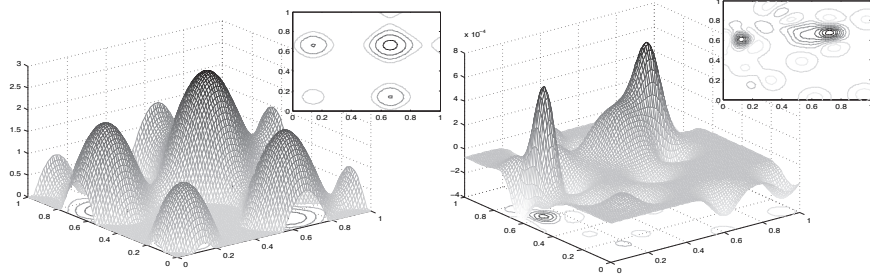

Figure 2: The 2D test function (left), and the estimate of the function based on the results of a typical run of 12 preference queries (right). The true function has eight local and one global maxima. The predictor identifies the region of the global maximum correctly and that of the local maxima less well, but requires far fewer queries than learning the entire function.

The probability of improvement at a point $\mathbf{x}^\star$ is simply given by a tail probability:

$$p(f^\star(\mathbf{x}^\star) \leq \mu_{\max}) = \Phi\left(\frac{\mu_{\max} - \mu(\mathbf{x}^\star)}{s(\mathbf{x}^\star)}\right),$$

where $f^\star(\mathbf{x}^\star) \sim \mathcal{N}(\mu(\mathbf{x}^\star), s^2(\mathbf{x}^\star))$. This statistical measure of improvement has been widely used in the field of experimental design and goes back many decades [Kushner, 1964]. However, it is known to be sensitive to the value of $\mu_{\max}$. To overcome this problem, [Jones *et al.*, 1998] defined the improvement over the current best point as $I(\mathbf{x}^\star) = \max\{0, \mu(\mathbf{x}^\star) - \mu_{\max}\}$, which resulted in an expected improvement of

$$EI(\mathbf{x}^\star) = \begin{cases} (\mu_{\max} - \mu(\mathbf{x}^\star))\Phi(d) + s(\mathbf{x}^\star)\phi(d) & \text{if } s > 0 \\ 0 & \text{if } s = 0 \end{cases}$$

where $d = \frac{\mu_{\max} - \mu(\mathbf{x}^\star)}{s(\mathbf{x}^\star)}$.

To find the point at which to sample, we still need to maximize the constrained objective $EI(\mathbf{x}^\star)$ over $\mathbf{x}^\star$. *Unlike the original unknown cost function, $EI(\cdot)$ can be cheaply sampled.* Furthermore, for the purposes of our application, it is not necessary to guarantee that we find the global maximum, merely that we can quickly locate a point that is likely to be as good as possible. The original EGO work used a branch-and-bound algorithm, but we found it was very difficult to get good bounds over large regions. Instead we use *DIRECT* [Jones *et al.*, 1993], a fast, approximate, derivative-free optimization algorithm, though we conjecture that for larger dimensional spaces, sequential quadratic programming with interior point methods might be a better alternative.

## 3 Experiments

The goal of our algorithm is to find a good approximation of the maximum of a latent function using preference queries. In order to measure our method's effectiveness in achieving this goal, we create a function $\mathbf{f}$ for which the optimum is known. At each time step, a query is generated in which two points $\mathbf{x}_1$ and $\mathbf{x}_2$ are adaptively selected, and the preference is found, where $\mathbf{f}(\mathbf{x}_1) > \mathbf{f}(\mathbf{x}_2) \Leftrightarrow \mathbf{x}_1 \succ \mathbf{x}_2$. After each preference, we measure the error, defined as $\epsilon = \mathbf{f}_{\max} - \mathbf{f}(\text{argmax}_x \mathbf{f}^*(\mathbf{x}))$, that is, the difference between the true maximum of $\mathbf{f}$ and the value of $\mathbf{f}$ at the point predicted to be the maximum. Note that by design, this does not penalize the algorithm for drawing samples from $\mathcal{X}$ that are far from $\text{argmax}_x$, or for predicting a latent function that differs from the true function. We are not trying to learn the entire valuation function, which would take many more queries – we seek only to maximize the valuation, which involves accurate modelling only in the areas of high valuation.

We measured the performance of our method on three functions – 2D, 4D and 6D. By way of demonstration, Figure 2 shows the actual 2D functions and the typical prediction after several queries. The test functions are defined as:

$$\mathbf{f}_{2d} = \max\{0, \sin(x_1) + x_1/3 + \sin(12x_1) + \sin(x_2) + x_2/3 + \sin(12x_2) - 1\}$$

$$\mathbf{f}_{4d,6d} = \sum_{i=1}^{d} \sin(x_i) + x_i/3 + \sin(12x_i)$$

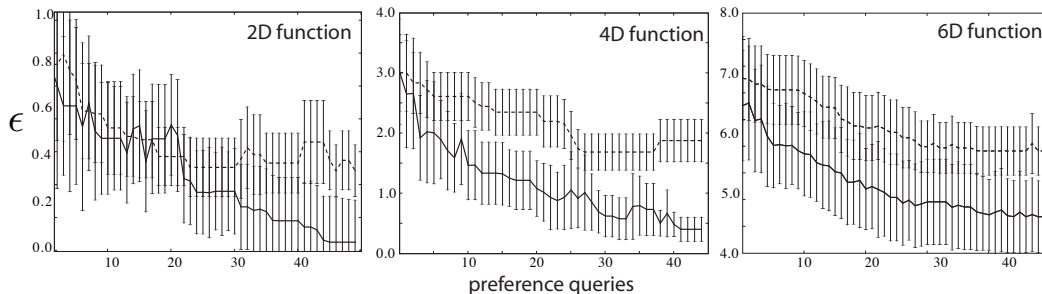

Figure 3: The evolution of error for the estimate of the optimum on the test functions. The plot shows the error evolution $\epsilon$ against the number of queries. The solid line is our method; the dashed is a baseline comparison in which each query point is selected randomly. The performance is averaged over 20 runs, with the error bars showing the variance of $\epsilon$.

all defined over the range $[0, 1]^d$. We selected these equations because they seem both general and difficult enough that we can safely assume that if our method works well on them, it should work on a large class of real-world problems — they have multiple local minima to get trapped in and varying landscapes and dimensionality. Unfortunately, there has been little work in the psychoperception literature to indicate what a good test function would be for our problem, so we have had to rely to an extent on our intuition to develop suitable test cases.

The results of the experiments are shown in Figure 3. In all cases, we simulate 50 queries using our method (here called $\max_{EI}$). As a baseline, we compare against 50 queries using the maximum variance of the model ($\max_s$), which is a common criterion in active learning for regression [Seo *et al.*, 2000; Chu and Ghahramani, 2005a]. We repeated each experiment 20 times and measured the mean and variance of the error evolution. We find that it takes far fewer queries to find a good result using $\max_{EI}$ in all cases. In the 2D case, for example, after 20 queries, $\max_{EI}$ already has better average performance than $\max_s$ achieves after 50, and in both the 2D and 4D scenarios, $\max_{EI}$ steadily improves until it find the optima, while $\max_s$ soon reaches a plateau, improving only slightly, if at all, while it tries to improve the global fit to the latent function. In the 6D scenario, neither algorithm succeeds well in finding the optimum, though $\max_{EI}$ clearly comes closer. We believe the problem is that in six dimensions, the space is too large to adequately explore with so few queries, and variance remains quite high throughout the space. We feels that requiring more than 50 user queries in a real application would be unacceptable, so we are instead currently investigating extensions that will allow the user to direct the search in higher dimensions.

## 4   Preference Gallery for Material Design

Properly modeling the appearance of a material is a necessary component of realistic image synthesis. The appearance of a material is formalized by the notion of the Bidirectional Reflectance Distribution Function (BRDF). In computer graphics, BRDFs are most often specified using various analytical models observing the physical laws of reciprocity and energy conservation while also exhibiting shadowing, masking and Fresnel reflectance phenomenon. Realistic models are therefore fairly complex with many parameters that need to be adjusted by the designer. Unfortunately these parameters can interact in non-intuitive ways, and small adjustments to certain settings may result in non-uniform changes in appearance. This can make the material design process quite difficult for the end user, who cannot expected to be an expert in the field of appearance modeling.

Our application is a solution to this problem, using a "preference gallery" approach, in which users are simply required to view two or more images rendered with different material properties and indicate which ones they prefer. To maximize the valuation, we use an implementation of the model described in Section 2. In practice, the first few examples will be points of high variance, since little of the space is explored (that is, the model of user valuation is very uncertain). Later samples tend to be in regions of high valuation, as a model of the user's interest is learned.

We use our active preference learning model on an example gallery application for helping users find a desired BRDF. For the purposes of this example, we limit ourselves to isotropic materials and ignore wavelength dependent effects in reflection. The gallery uses the Ashikhmin-Shirley Phong

Table 1: Results of the user study

| algorithm | trials | $n$ (mean $\pm$ std) |
|---|---|---|
| latin hypercubes | 50 | $18.40 \pm 7.87$ |
| $\max_s$ | 50 | $17.87 \pm 8.60$ |
| $\max_{EI}$ | 50 | $8.56 \pm 5.23$ |

model [Ashikhmin and Shirley, 2000] for the BRDFs which was recently validated to be well suited for representing real materials [Ngan *et al.*, 2005]. The BRDFs are rendered on a sphere under high frequency natural illumination as this has been shown to be the desired setting for human preception of reflectance [Fleming *et al.*, 2001]. Our gallery demonstration presents the user with two BRDF images at a time. We start with four predetermined queries to "seed" the parameter space, and after that use the learned model to select gallery images. The GP model is updated after each preference is indicated. We use parameters of real measured materials from the MERL database [Ngan *et al.*, 2005] for seeding the parameter space, but can draw arbitrary parameters after that.

## 4.1 User Study

To evaluate the performance of our application, we have run a simple user study in which the generated images are restricted to a subset of 38 materials from the MERL database that we deemed to be representative of the appearance space of the measured materials. The user is given the task of finding a single randomly-selected image from that set by indicating preferences. Figure 4 shows a typical user run, where we ask the user to use the preference gallery to find a provided target image. At each step, the user need only indicate the image they think looks most like the target. This would, of course, be an unrealistic scenario if we were to be evaluating the application from an HCI stance, but here we limit our attention to the model, where we are interested here in demonstrating that with human users maximizing valuation is preferable to learning the entire latent function.

Using five subjects, we compared 50 trials using the EIF to select the images for the gallery ($\max_{EI}$), 50 trials using maximum variance ($\max_s$, the same criterion as in the experiments of Section 3), and 50 trials using samples selected using a randomized Latin hypercube algorithm. In each case, one of the gallery images was the image with the highest predicted valuation and the other was selected by the algorithm. The algorithm type for each trial was randomly selected by the computer and neither the experimenter nor the subjects knew which of the three algorithms was selecting the images. The results are shown in Table 1. $n$ is the number clicks required of the user to find the target image. Clearly $\max_{EI}$ dominates, with a mean $n$ less than half that of the competing algorithms. Interestingly, selecting images using maximum variance does not perform much better than random. We suspect that this is because $\max_s$ has a tendency to select images from the corners of the parameter space, which adds limited information to the other images, whereas Latin hypercubes at least guarantees that the selected images fill the space.

Active learning is clearly a powerful tool for situations where human input is required for learning. With this paper, we have shown that understanding the task — and exploiting the quirks of human cognition — is also essential if we are to deploy real-world active learning applications. As people come to expect their machines to act intelligently and deal with more complex environments, machine learning systems that can collaborate with users and take on the tedious parts of users' cognitive burden has the potential to dramatically affect many creative fields, from business to the arts to science.

## References

[Ashikhmin and Shirley, 2000]  M. Ashikhmin and P. Shirley.  An anisotropic phong BRDF model. *J. Graph. Tools*, 5(2):25–32, 2000.

[Chu and Ghahramani, 2005a]  W. Chu and Z. Ghahramani.  Extensions of Gaussian processes for ranking: semi-supervised and active learning. In *Learning to Rank workshop at NIPS-18*, 2005.

[Chu and Ghahramani, 2005b]  W. Chu and Z. Ghahramani.  Preference learning with Gaussian processes.  In *ICML*, 2005.

[Cohn *et al.*, 1996]  D. A. Cohn, Z. Ghahramani, and M. I. Jordan.  Active learning with statistical models. *Journal of Artificial Intelligence Research*, 4:129–145, 1996.

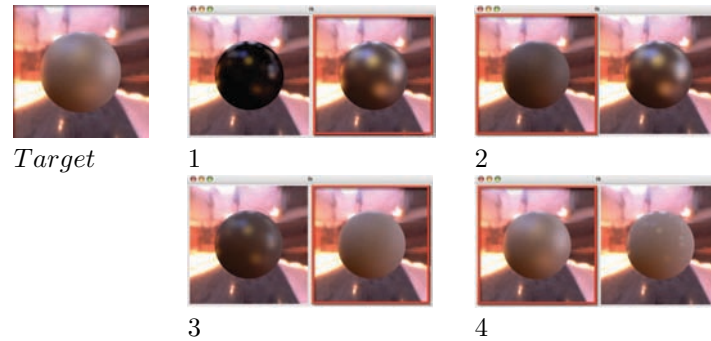

Figure 4: A shorter-than-average but otherwise typical run of the preference gallery tool. At each (numbered) iteration, the user is provided with two images generated with parameter instances and indicates the one they think most resembles the target image (top-left) they are looking for. The boxed images are the user's selections at each iteration.

[Cooper *et al.*, 2007] S. Cooper, A. Hertzmann, and Z. Popović. Active learning for motion controllers. In *SIGGRAPH*, 2007.

[Élő, 1978] Á. Élő. *The Rating of Chess Players: Past and Present*. Arco Publishing, New York, 1978.

[Fleming *et al.*, 2001] R. Fleming, R. Dror, and E. Adelson. How do humans determine reflectance properties under unknown illumination? In *CVPR Workshop on Identifying Objects Across Variations in Lighting*, 2001.

[Glickman and Jensen, 2005] M. E. Glickman and S. T. Jensen. Adaptive paired comparison design. *Journal of Statistical Planning and Inference*, 127:279–293, 2005.

[Guestrin *et al.*, 2005] C. Guestrin, A. Krause, and A. P. Singh. Near-optimal sensor placements in Gaussian processes. In *Proceedings of the 22nd International Conference on Machine Learning (ICML-05)*, 2005.

[Jones *et al.*, 1993] D. R. Jones, C. D. Perttunen, and B. E. Stuckman. Lipschitzian optimization without the Lipschitz constant. *J. Optimization Theory and Apps*, 79(1):157–181, 1993.

[Jones *et al.*, 1998] D. R. Jones, M. Schonlau, and W. J. Welch. Efficient global optimization of expensive black-box functions. *J. Global Optimization*, 13(4):455–492, 1998.

[Kendall, 1975] M. Kendall. *Rank Correlation Methods*. Griffin Ltd, 1975.

[Kingsley, 2006] D. C. Kingsley. Preference uncertainty, preference refinement and paired comparison choice experiments. Dept. of Economics, University of Colorado, 2006.

[Kushner, 1964] H. J. Kushner. A new method of locating the maximum of an arbitrary multipeak curve in the presence of noise. *Journal of Basic Engineering*, 86:97–106, 1964.

[Marks *et al.*, 1997] J. Marks, B. Andalman, P. A. Beardsley, W. Freeman, S. Gibson, J. Hodgins, T. Kang, B. Mirtich, H. Pfister, W. Ruml, K. Ryall, J. Seims, and S. Shieber. Design galleries: A general approach to setting parameters for computer graphics and animation. *Computer Graphics*, 31, 1997.

[McFadden, 2001] D. McFadden. Economic choices. *The American Economic Review*, 91:351–378, 2001.

[Mosteller, 1951] F. Mosteller. Remarks on the method of paired comparisons: I. the least squares solution assuming equal standard deviations and equal correlations. *Psychometrika*, 16:3–9, 1951.

[Ngan *et al.*, 2005] A. Ngan, F. Durand, and W. Matusik. Experimental analysis of BRDF models. In *Proceedings of the Eurographics Symposium on Rendering*, pages 117–226, 2005.

[Ngan *et al.*, 2006] A. Ngan, F. Durand, and W. Matusik. Image-driven navigation of analytical BRDF models. In T. Akenine-Möller and W. Heidrich, editors, *Eurographics Symposium on Rendering*, 2006.

[Sasena, 2002] M. J. Sasena. *Flexibility and Efficiency Enhancement for Constrained Global Design Optimization with Kriging Approximations*. PhD thesis, University of Michigan, 2002.

[Seo *et al.*, 2000] S. Seo, M. Wallat, T. Graepel, and K. Obermayer. Gaussian process regression: active data selection and test point rejection. In *Proceedings of IJCNN 2000*, 2000.

[Siegel and Castellan, 1988] S. Siegel and N. J. Castellan. *Nonparametric Statistics for the Behavioral Sciences*. McGraw-Hill, 1988.

[Stern, 1990] H. Stern. A continuum of paired comparison models. *Biometrika*, 77:265–273, 1990.

[Thurstone, 1927] L. Thurstone. A law of comparative judgement. *Psychological Review*, 34:273–286, 1927.

[Tong and Koller, 2000] S. Tong and D. Koller. Support vector machine active learning with applications to text classification. In *Proc. ICML-00*, 2000.

